# Neural Network - Gaussian Mixture Hybrid for Speech Recognition or Density Estimation

**Yoshua Bengio**
Dept. Brain and Cognitive Sciences
Massachusetts Institute of Technology
Cambridge, MA 02139

**Renato De Mori**
School of Computer Science
McGill University
Canada

**Giovanni Flammia**
Speech Technology Center,
Aalborg University, Denmark

**Ralf Kompe**
Erlangen University, Computer Science
Erlangen, Germany

## Abstract

The subject of this paper is the integration of multi-layered Artificial Neural Networks (ANN) with probability density functions such as Gaussian mixtures found in continuous density Hidden Markov Models (HMM). In the first part of this paper we present an ANN/HMM hybrid in which all the parameters of the the system are simultaneously optimized with respect to a single criterion. In the second part of this paper, we study the relationship between the density of the inputs of the network and the density of the outputs of the networks. A few experiments are presented to explore how to perform density estimation with ANNs.

## 1   INTRODUCTION

This paper studies the integration of Artificial Neural Networks (ANN) with probability density functions (pdf) such as the Gaussian mixtures often used in continuous density Hidden Markov Models. The ANNs considered here are multi-layered or recurrent networks with hyperbolic tangent hidden units. Raw or preprocessed data is fed to the ANN, and the outputs of the ANN are used as observations for a parametric probability density function such as a Gaussian mixture. One may view either the ANN as an adaptive preprocessor for the Gaussian mixture, or the Gaussian mixture as a statistical postprocessor for the ANN. A useful role for the ANN would be to transform the input data so that it can be more efficiently modeled by a Gaussian mixture. An interesting situation is one in which most of the input data points can be described in a lower dimensional space. In this case, it is desired that the ANN learns the possibly non-linear transformation to a more compact representation.

In the first part of this paper, we briefly describe a hybrid of ANNs and Hidden Markov Models (HMM) for continuous speech recognition. More details on this system can be found in (Bengio 91). In this hybrid, all the free parameters are simultaneously optimized with respect to a single criterion. In recent years, many related combinations have been studied (e.g., Levin 90, Bridle 90, Bourlard & Wellekens 90). These approaches are often motivated by observed advantages and disadvantages of ANNs and HMMs in speech recognition (Bourlard & Wellekens 89, Bridle 90). Experiments of phoneme recognition on the TIMIT database with the proposed ANN/HMM hybrid are reported. The task under study is the recognition (or spotting) of plosive sounds in continuous speech. Comparative results on this task show that the hybrid performs better than the ANN alone, better than the ANN followed by a dynamic programming based postprocessor using duration constraints, and better than the HMM alone. Furthermore, a global optimization of all the parameters of the system also yielded better performance than a separate optimization.

In the second part of this paper, we attempt to extend some of the findings of the first part, in order to use the same basic architecture (ANNs followed by Gaussian mixtures) to perform density estimation. We establish the relationship between the network input and output densities, and we then describe a few experiments exploring how to perform density estimation with this system.

## 2    ANN/HMM HYBRID

In a HMM, the likelihood of the observations, given the model, depends in a simple continuous way on the observations. It is therefore possible to compute the derivative of an optimization criterion $C$, with respect to the observations of the HMM. For example, one may use the criterion of the Maximum Likelihood (ML) of the observations, or of the Maximum Mutual Information (MMI) between the observations and the correct sequence. If the observation at each instant is the vector output, $Y_t$, of an ANN, then one can use this gradient, $\frac{\partial C}{\partial Y_t}$, to optimize the parameters of the ANN with back-propagation. See (Bridle 90, Bottou 91, Bengio 91, Bengio et al 92) on ways to compute this gradient.

### 2.1    EXPERIMENTS

A preliminary experiment has been performed using a prototype system based on the integration of ANNs with HMMs. The ANN was initially trained based on a prior task decomposition. The task is the recognition of plosive phonemes pronounced by a large speaker population. The 1988 version of the TIMIT continuous speech database has been used for this purpose. SI and SX sentences from regions 2, 3 and 6 were used, with 1080 training sentences and 224 test sentences, 135 training speakers and 28 test speakers. The following 8 classes have been considered: /p/,/t/,/k/,/b/,/d/,/g/,/dx/,/all other phones/. Speaker-independent recognition of plosive phonemes in continuous speech is a particularly difficult task because these phonemes are made of short and non-stationary events that are often confused with other acoustically similar consonants or may be merged with other unit segments by a recognition system.

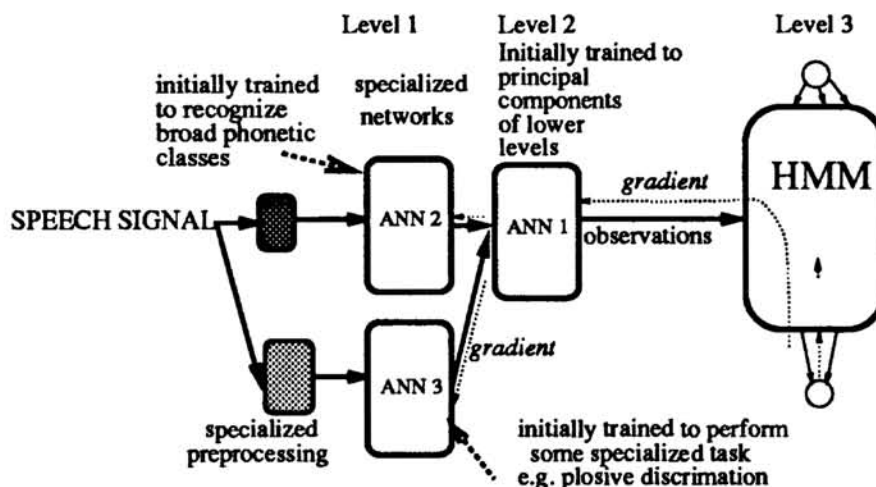

Figure 1: *Architecture of the ANN/HMM Hybrid for the Experiments.*

The ANNs were trained with back-propagation and on-line weight update. As discussed in (Bengio 91), speech knowledge is used to design the input, output, and architecture of the system and of each one of the networks. The experimental system is based on the scheme shown in Figure 1. The architecture is built on three levels. The approach that we have taken is to select different input parameters and different ANN architectures depending on the phonetic features to be recognized. At level 1, two ANNs are initially trained to perform respectively plosive recognition (ANN3) and broad classification of phonemes (ANN2). ANN3 has delays and recurrent connections and is trained to recognize static articulatory features of plosives in a way that depends of the place of articulation of the right context phoneme. ANN2 has delays but no recurrent connections. The design of ANN2 and ANN3 is described in more details in (Bengio 91). At level 2, ANN1 acts as an integrator of parameters generated by the specialized ANNs of level 1. ANN1 is a linear network that initially computes the 8 principal components of the concatenated output vectors of the lower level networks (ANN2 and ANN3). In the experiment described below, the combined network (ANN1+ANN2+ANN3) has 23578 weights. Level 3 contains the HMMs, in which each distribution is modeled by a Gaussian mixture with 5 densities. See (Bengio et al 92) for more details on the topology of the HMM. The covariance matrix is assumed to be diagonal since the observations are initially principal components and this assumption reduces significantly the number of parameters to be estimated. After one iteration of ML re-estimation of the HMM parameters only, all the parameters of the hybrid system were simultaneously tuned to maximize the ML criterion for the next 2 iterations. Because of the simplicity of the implementation of the hybrid trained with ML, this criterion was used in these experiments. Although such an optimization may theoretically worsen performance[1], we observed an marked improvement in performance after the final global tuning. This may be explained by the fact that a nearby local maximum of

the likelihood is attained from the initial starting point based on prior and separate training of the ANN and the HMM.

Table 1: *Comparative Recognition Results. % recognized = 100 - % substitutions - % deletions. % accuracy = 100 - % substitutions - % deletions -% insertions.*

|                     | % rec | % ins | % del | % subs | % acc |
|---------------------|-------|-------|-------|--------|-------|
| ANNs alone          | 85    | 32    | 0.04  | 15     | 53    |
| HMMs alone          | 76    | 6.3   | 2.2   | 22.3   | 69    |
| ANNs+DP             | 88    | 16    | 0.01  | 11     | 72    |
| ANNs+HMM            | 87    | 6.8   | 0.9   | 12     | 81    |
| ANNs+HMM+global opt.| 90    | 3.8   | 1.4   | 9.0    | 86    |

In order to assess the value of the proposed approach as well as the improvements brought by the HMM as a post-processor for time alignment, the performance of the hybrid system was evaluated and compared with that of a simple post-processor applied to the outputs of the ANNs and with that of a standard dynamic programming postprocessor that models duration probabilities for each phoneme. The simple post-processor assigns a symbol to each output frame of the ANNs by comparing the target output vectors with actual output vectors. It then smoothes the resulting string to remove very short segments and merges consecutive segments that have the same symbol. The dynamic programming (DP) postprocessor finds the sequence of phones that minimizes a cost that imposes durational constraints for each phoneme. In the HMM alone system, the observations are the cepstrum and the energy of the signal, as well as their derivatives. Comparative results for the three systems are summarized in Table 1.

## 3  DENSITY ESTIMATION WITH AN ANN

In this section, we consider an extension of the system of the previous section. The objective is to perform density estimation of the inputs of the ANN. Instead of maximizing a criterion that depends on the density of the outputs of an ANN, we maximize the likelihood of inputs of the ANN. Hence the ANN is more than a preprocessor for the gaussian mixtures, it is part of the probability density function that is to be estimated. Instead of representing a pdf only with a set of spatially local functions or kernels such as gaussians (Silverman 86), we explore how to use a global transformation such as one performed by an ANN in order to represent a pdf. Let us first define some notation: $f_X(x) \stackrel{\text{def}}{=} p(X = x)$, $f_Y(y) \stackrel{\text{def}}{=} p(Y = y)$, and $f_{X|Y(X)}(x) \stackrel{\text{def}}{=} p(X = x \mid Y = y(x))$.

### 3.1  RELATION BETWEEN INPUT PDF AND OUTPUT PDF

**Theorem** *Suppose a random variable $Y$ (e.g., the outputs of an ANN) is a deterministic parametric function $y(X)$ of a random variable $X$ (here, the inputs of the ANN), where $y$ and $x$ are vectors of dimension $n_y$ and $n_x$. Let $J = \frac{\partial(y_1, y_2, \ldots y_{n_y})}{\partial(x_1, x_2, \ldots x_{n_x})}$*

---

not the outputs of the network.

*be the Jacobian of the transformation from $X$ to $Y$, and assume $J = UDV^t$ be a singular value decomposition of $J$, with $s(x) = | \prod_i^{n_y} D_{ii} |$ the product of the singular values. Suppose $Y$ is modeled by a probability density function $f_Y(y)$. Then, for $n_x >= n_y$ and $s(x) > 0$*

$$f_X(x) \;=\; f_Y(y(x)) \;\; f_{X|Y(X)}(x) \;\; s(x) \tag{1}$$

Proof. In the case in which $n_x = n_y$, by change of variable $y \to x$ in the following integral,

$$\int_{\Omega_y} f_Y(y)\, dy \;=\; 1 \tag{2}$$

we obtain the following result[2]:

$$f_X(x) = f_Y(y(x)) \; | \operatorname{Determinant}(J) | \tag{3}$$

Let us now consider the case $n_y < n_x$, i.e., the network has less outputs than inputs. In order to do so we will introduce an intermediate transformation to a space $Z$ of dimension $n_x$ in which some dimensions directly correspond to $Y$. Define $Z$ such that $\frac{\partial(z_1, z_2, \ldots, z_{n_x})}{\partial(x_1, x_2, \ldots, x_{n_x})} = V^t$. Decompose $Z$ into $Z'$ and $Z''$:

$$z' = (z_1, \ldots, z_{n_y}) \,, \;\; z'' = (z_{n_y+1}, \ldots, z_{n_x}) \tag{4}$$

There is a one-to-one mapping $y_z(z')$ between $Z'$ and $Y$, and its Jacobian is $UD'$, where $D'$ is the matrix composed of the first $n_y$ columns of $D$. Perform a change of variables $y \to z'$ in the integral of equation 2:

$$\int_{\Omega_{z'}} f_Y(y_z(z'))\, s\, dz' \;=\; 1 \tag{5}$$

In order to make a change of variable to the variable $x$, we have to specify the conditional pdf $f_{X|Y(X)}(x)$ and the corresponding pdf $p(z'' \mid z') = p(z'', z' \mid z') =^3 p(z \mid y) =^4 f_{X|Y(X)}(x)$. Hence we can write

$$\int_{\Omega_{z''}} p(z'' \mid z')\, dz'' \;=\; 1 \tag{6}$$

Multiplying the two integrals in equations 5 and 6, we obtain the following:

$$1 = \int_{\Omega_{z''}} p(z'' \mid z')\, dz'' \int_{\Omega_{z'}} f_Y(y_z(z'))\, s\, dz' = \int_{\Omega_z} f_Y(y_z(z')\, p(z'' \mid z')s\, dz \tag{7}$$

and substituting $z \to V^t x$:

$$\int_{\Omega_x} f_Y(y(x))\, f_{X|Y(X)}(x)\, s(x)\, dx \;=\; 1, \tag{8}$$

which yields to the general result of equation 1 $\square$.

Unfortunately, it is not clear how to efficiently evaluate $f_{X|Y(X)}(x)$ and then compute its derivative with respect to the network weights. In the experiments described in the next section we first study empirically the simpler case in which $n_x = n_y$.

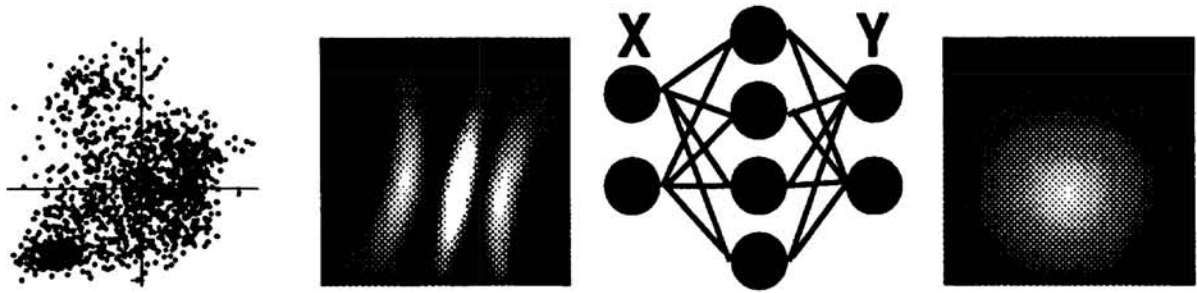

Figure 2: *First Series of Experiments on Density Estimation with an ANN, for data generated on a non-linear input curve. From left to right: Input samples, density of the input, X, estimated with ANN+Gaussian, ANN that maps X to Y, density of the output, Y, as estimated by a Gaussian.*

## 3.2   ESTIMATION OF THE PARAMETERS

When estimating a pdf, one can approximate the functions $f_Y(y)$ and $y(x)$ by parameterized functions. For example, we consider for the output pdf the class of densities $f_Y(y; \theta)$ modeled by a Gaussian mixture of a certain number of components, where $\theta$ is a set of means, variances and mixing proportions. For the non-linear transformation $y(x; \omega)$ from $X$ to $Y$, we choose an ANN, defined by its architecture and the values of its weights $\omega$. In order to choose values for the Gaussian and ANN parameters one can maximize the a-posteriori (MAP) probability of these parameters given the data, or if no prior is known or assumed, maximize the likelihood (ML) of the input data given the parameters. In the preliminary experiments described here, the logarithm of the likelihood of the data was maximized, i.e., the optimal parameters are defined as follows:

$$(\hat{\theta}, \hat{\omega}) = \underset{(\theta, \omega)}{\operatorname{argmax}} \sum_{x \in \Xi} \log(f_X(x)) \qquad (9)$$

where $\Xi$ is the set of inputs samples.

In order to estimate a density with the above described system, one computes the derivative of $p(X = x \mid \theta, \omega)$ with respect to $\omega$. If the output pdf is a Gaussian mixture, we reestimate its parameters $\theta$ with the EM algorithm (only $f_Y(y)$ depends on $\theta$ in the expression for $f_X(x)$ in equations 3 or 1). Differentiating equation 3 with respect to $\omega$ yields:

$$\frac{\partial}{\partial \omega}(\log f_X(x)) = \frac{\partial}{\partial \omega}(\log f_Y(y(x; \omega); \theta)) + \sum_{i,j} \frac{\partial}{\partial J_{ij}}(\log(\text{Determinant}(J)))\frac{\partial J_{ij}}{\partial \omega}$$

$$(10)$$

The derivative of the logarithm of the determinant can be computed simply as follows (Bottou 91):

$$\frac{\partial}{\partial J_{ij}}(\log(\text{Determinant}(J))) = (J^{-1})_{ji}, \qquad (11)$$

since $\forall A$, $\text{Determinant}(A) = \sum_j A_{ij}\text{Cofactor}_{ij}(A)$ , and $(A^{-1})_{ij} = \frac{\text{Cofactor}_{ji}(A)}{\text{Determinant}(A)}$.

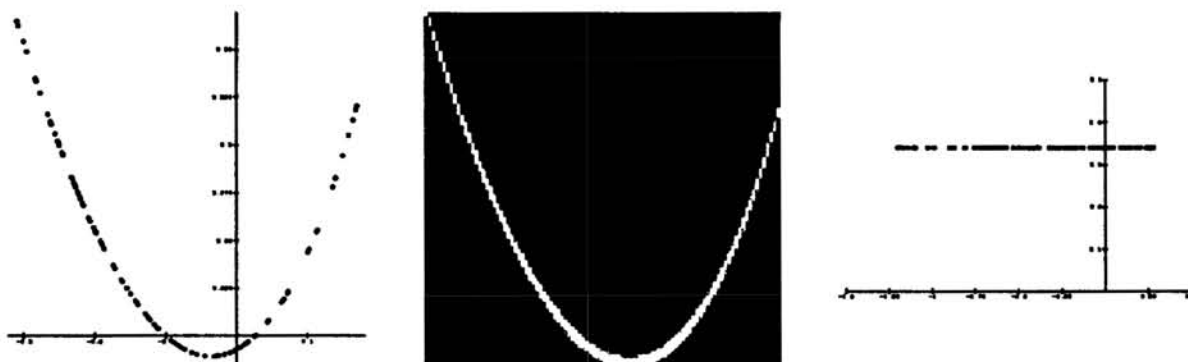

Figure 3: *Second Series of Experiments on Density Estimation with an ANN. From left to right: Input samples, density with non-linear net + Gaussian, output samples after network transformation.*

## 3.3  EXPERIMENTS

The first series of experiments verified that a transformation of the inputs with an ANN could improve the likelihood of the inputs and that gradient ascent in the ML criterion could find a good solution. In these experiments, we attempt to model some two-dimensional data extracted from a speech database. The 1691 training data points are shown in the left of Figure 2. In the first experiment, a diagonal Gaussian is used, with no ANN. In the second experiment a linear network and a diagonal Gaussian are used. In the third experiment, a non-linear network with 4 hidden units and a diagonal Gaussian are used. The average log likelihoods obtained on a test set of 617 points were -3.00, -2.95 and -2.39 respectively for the three experiments. The estimated input and output pdfs for the last experiment are depicted in Figure 2, with white indicating high density and black low density.

The second series of experiments addresses the following question: if we use a Gaussian mixture with diagonal covariance matrix and most of the data is on a non-linear hypersurface $\Phi$ of dimension less than $n_x$, can the ANN's outputs separate the dimensions in which the data varies greatly (along $\Phi$) from those in which it almost doesn't (orthogonal to $\Phi$)? Intuitively, it appears that this will be the case, because the variance of outputs which don't vary with the data will be close to zero, while the determinant of the Jacobian is non-zero. The likelihood will correspondingly tend to infinity. The first experiment in this series verified that this was the case for linear networks. For data generated on a diagonal line in 2-dimensional space, the resulting network separated the "variant" dimension from the "invariant" dimension, with one of the output dimensions having near zero variance, and the transformed data lying on a line parallel to the other output dimension.

Experiments with non-linear networks suggest that with such networks, a solution that separates the variant dimensions from the invariant ones is not easily found by gradient ascent. However, it was possible to show that such a solution was at a maximum (possibly local) of the likelihood. A last experiment was designed to demonstrate this. The input data, shown in Figure 3, was artificially generated to make sure that a solution existed. The network had 2 inputs, 3 hidden units and 2

outputs. The input samples and the input density corresponding to the weights in a maximum of the likelihood are displayed in Figure 3, along with the transformed input data for those weights. The points are projected by the ANN to a line parallel to the first output dimension. Any variation of the weights from that solution, in the direction of the gradient, even with a learning rate as small as $10^{-14}$, yielded either no perceptible improvement or a decrease in likelihood.

## 4    CONCLUSION

This paper has studied an architecture in which an ANN performs a non-linear transformation of the data to be analyzed, and the output of the ANN is modeled by a Gaussian mixture. The design of the ANN can incorporate prior knowledge about the problem, for example to modularize the task and perform an initial training of the sub-networks. In phoneme recognition experiments, an ANN/HMM hybrid based on this architecture performed better than the ANN alone or the HMM alone. In the second part of th paper, we have shown how the pdf of the input of the network relates to the pdf of the outputs of the network. The objective of this work is to perform density estimation with a non-local non-linear transformation of the data. Preliminary experiments showed that such estimation was possible and that it did improve the likelihood of the resulting pdf with respect to using only a Gaussian pdf. We also studied how this system could perform a non-linear analogue to principal components analysis.

**References**

Bengio Y. 1991.  Artificial Neural Networks and their Application to Sequence Recognition. PhD Thesis, School of Computer Science, McGill University, Montreal, Canada.

Bengio Y., De Mori R., Flammia G., and Kompe R. 1992.  Phonetically motivated acoustic parameters for continuous speech recognition using artificial neural networks. To appear in *Speech Communication*.

Bottou L. 1991.  Une approche théorique à l'apprentissage connexioniste; applications à la reconnaissance de la parole. Doctoral Thesis, Université de Paris Sud, France.

Bourlard, H. and Wellekens, C.J. (1989). Speech pattern discrimination and multilayer perceptrons. *Computer, Speech and Language*, vol. 3, pp. 1-19.

Bridle J.S. 1990. Training stochastic model recognition algorithms as networks can lead to maximum mutual information estimation of parameters. *Advances in Neural Information Processing Systems 2*, (ed. D.S. Touretzky) Morgan Kauffman Publ., pp. 211-217.

Levin E. 1990. Word recognition using hidden control neural architecture. *Proceedings of the International Conference on Acoustics, Speech and Signal Processing*, Albuquerque, NM, April 90, pp. 433-436.

Silverman B.W. 1986. Density Estimation for Statistics and Data Analysis. Chapman and Hall, New York, NY.

## Footnotes

[1]In section 3, we consider maximization of the likelihood of the inputs of the network,

[2] in that case, $| \operatorname{Determinant}(J) | = s$ and $f_{X|Y(X)}(x) = 1$.

[3] knowing $z'$ is equivalent to knowing $y$.

[4] because $z = V^t x$ and $\operatorname{Determinant}(V) = 1$.
